# Bayesian Network Induction via Local Neighborhoods

**Dimitris Margaritis**
Department of Computer Science
Carnegie Mellon University
Pittsburgh, PA 15213
*D.Margaritis@cs.cmu.edu*

**Sebastian Thrun**
Department of Computer Science
Carnegie Mellon University
Pittsburgh, PA 15213
*S.Thrun@cs.cmu.edu*

## Abstract

In recent years, Bayesian networks have become highly successful tool for diagnosis, analysis, and decision making in real-world domains. We present an efficient algorithm for learning Bayes networks from data. Our approach constructs Bayesian networks by first identifying each node's Markov blankets, then connecting nodes in a maximally consistent way. In contrast to the majority of work, which typically uses hill-climbing approaches that may produce dense and causally incorrect nets, our approach yields much more compact causal networks by heeding independencies in the data. Compact causal networks facilitate fast inference and are also easier to understand. We prove that under mild assumptions, our approach requires time polynomial in the size of the data and the number of nodes. A randomized variant, also presented here, yields comparable results at much higher speeds.

## 1 Introduction

A great number of scientific fields today benefit from being able to automatically estimate the probability of certain quantities of interest that may be difficult or expensive to observe directly. For example, a doctor may be interested in estimating the probability of heart disease from indications of high blood pressure and other directly measurable quantities. A computer vision system may benefit from a probability distribution of buildings based on indicators of horizontal and vertical straight lines. Probability densities proliferate the sciences today and advances in its estimation are likely to have a wide impact on many different fields.

Bayesian networks are a succinct and efficient way to represent a joint probability distribution among a set of variables. As such, they have been applied to fields such as those mentioned [Herskovits90][Agosta88]. Besides their ability for density estimation, their semantics lend them to what is sometimes loosely referred to as *causal discovery*, namely directional relationships among quantities involved. It has been widely accepted that the most parsimonious representation for a Bayesian net is one that closely represents the causal independence relationships that may exist. For these reasons, there has been great interest in automatically inducing the structure of Bayesian nets automatically from data, preferably also preserving the independence relationships in the process.

Two research approaches have emerged. The first employs independence properties of the underlying network that produced the data in order to discover parts of its structure. This approach is mainly exemplified by the SGS and PC algorithms in [Spirtes93], as well

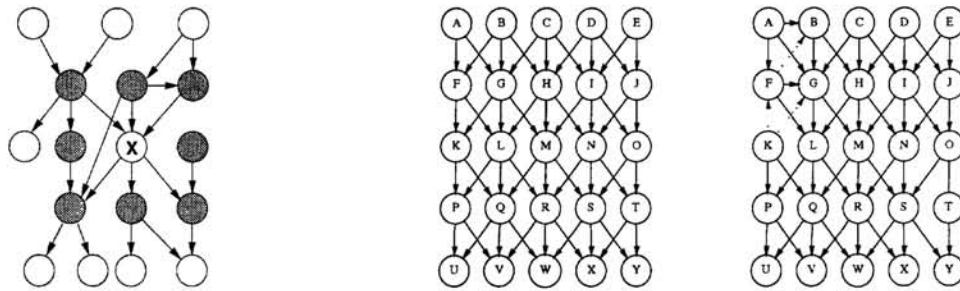

Figure 1: On the left, an example of a Markov blanket of variable $X$ is shown. The members of the blanket are shown shaded. On the right, an example reconstruction of a $5 \times 5$ rectangular net of branching factor 3 by the algorithm presented in this paper using 20000 samples. Indicated by dotted lines are 3 directionality errors.

as for restricted classes such as trees [Chow68] and polytrees [Rebane87]. The second approach is concerned more with data prediction, disregarding independencies in the data. It is typically identified with a greedy hill-climbing or best-first beam search in the space of legal structures, employing as a scoring function a form of data likelihood, sometimes penalized for network complexity. The result is a local maximum score network structure for representing the data, and is one of the more popular techniques used today.

This paper presents an approach that belongs in the first category. It addresses the two main shortcomings of the prior work which, we believe, are preventing its use from becoming more widespread. These two disadvantages are: exponential execution times, and proneness to errors in dependence tests used. The former problem is addressed in this paper in two ways. One is by identifying the local neighborhood of each variable in the Bayesian net as a preprocessing step, in order to facilitate the recovery of the local structure around each variable in polynomial time under the assumption of bounded neighborhood size. The second, randomized version goes one step further, employing a user-specified number of randomized tests (constant or logarithmic) in order to ascertain the same result with high probability. The second disadvantage of this research approach, namely proneness to errors, is also addressed by the randomized version, by using multiple data sets (if available) and Bayesian accumulation of evidence.

## 2    The Grow-Shrink Markov Blanket Algorithm

The concept of the *Markov blanket* of a variable or a set of variables is central to this paper. The concept itself is not new. For example, see [Pearl88]. It is surprising, however, how little attention it has attracted for all its being a fundamental property of a Bayesian net. What is new in this paper is the introduction of the explicit use of this idea to effectively limit unnecessary computation, as well as a simple algorithm to compute it. The definition of a Markov blanket is as follows: denoting $\mathbf{V}$ as the set of variables and $X \leftrightarrow_{\mathbf{S}} Y$ as the conditional dependence of $X$ and $Y$ given the set $\mathbf{S}$, the Markov blanket $\mathbf{BL}(X) \subseteq \mathbf{V}$ of $X \in \mathbf{V}$ is any set of variables such that for any $Y \in \mathbf{V} - \mathbf{BL}(X) - \{X\}$, $X \not\leftrightarrow_{\mathbf{BL}(X)} Y$. In other words, $\mathbf{BL}(X)$ completely shields variable $X$ from any other variable in $\mathbf{V}$. The notion of a *minimal Markov blanket*, called a *Markov boundary*, is also introduced in [Pearl88] and its uniqueness shown under certain conditions. The Markov boundary is not unique in certain pathological situations, such as the equality of two variables. In our following discussion we will assume that the conditions necessary for its existence and uniqueness are satisfied and we will identify the Markov blanket with the Markov boundary, using the notation $\mathbf{B}(X)$ for the blanket of variable $X$ from now on. It is also illuminating to mention that, in the Bayesian net framework, the Markov blanket of a node $X$ is easily identifiable from the graph: it consists of all parents, children and parents of children of $X$. An example Markov blanket is shown in Fig. 1. Note that any of these nodes, say $Y$, is dependent with $X$ given $\mathbf{B}(X) - \{Y\}$.

---

1. $\mathbf{S} \leftarrow \emptyset$.

2. While $\exists Y \in \mathbf{V} - \{X\}$ such that $Y \leftrightarrow_{\mathbf{S}} X$, do $\mathbf{S} \leftarrow \mathbf{S} \cup \{Y\}$.    **[Growing phase]**

3. While $\exists Y \in \mathbf{S}$ such that $Y \not\leftrightarrow_{\mathbf{S}-\{Y\}} X$, do $\mathbf{S} \leftarrow \mathbf{S} - \{Y\}$.    **[Shrinking phase]**

4. $\mathbf{B}(X) \leftarrow \mathbf{S}$.

---

Figure 2: The basic Markov blanket algorithm.

The algorithm for the recovery of the Markov blanket of $X$ is shown in Fig. 2. The idea behind step 2 is simple: as long as the Markov blanket property of $X$ is violated (*ie.* there exists a variable in $\mathbf{V}$ that is dependent on $X$), we add it to the current set $\mathbf{S}$ until there are no more such variables. In this process however, there may be some variables that were added to $\mathbf{S}$ that were really outside the blanket. Such variables would have been rendered independent from $X$ at a later point when "intervening" nodes of the underlying Bayesian net were added to $\mathbf{S}$. This observation necessitates step 3, which identifies and removes those variables. The algorithm is efficient, requiring only $O(n)$ conditional tests, making its running time $O(n\,|\mathbf{D}|)$, where $n = |\mathbf{V}|$ and $\mathbf{D}$ is the set of examples. For a detailed derivation of this bound as well as a formal proof of correctness, see [Margaritis99]. In practice one may try to minimize the number of tests in step 3 by heuristically ordering the variables in the loop of step 2, for example by ascending mutual information or probability of dependence between $X$ and $Y$ (as computed using the $\chi^2$ test, see section 5).

## 3   Grow-Shrink (GS) Algorithm for Bayesian Net Induction

The recovery of the local structure around each node is greatly facilitated by the knowledge of the nodes' Markov blankets. What would normally be a daunting task of employing dependence tests conditioned on an exponential number of subsets of large sets of variables—even though most of their members may be irrelevant—can now be focused on the Markov blankets of the nodes involved, making structure discovery much faster and more reliable. We present below the plain version of the GS algorithm that utilizes blanket information for inducing the structure of a Bayesian net. At a later point of this paper, we will present a robust, randomized version that has the potential of being faster and more reliable, as well as being able to operate in an "anytime" manner.

In the following $\mathbf{N}(X)$ represents the direct neighbors of $X$.

**[ Compute Markov Blankets ]**
    For all $X \in \mathbf{V}$, compute the Markov blanket $\mathbf{B}(X)$.

**[ Compute Graph Structure ]**
    For all $X \in \mathbf{V}$ and $Y \in \mathbf{B}(X)$, determine $Y$ to be a direct neighbor of $X$ if $X$ and $Y$ are dependent given $\mathbf{S}$ for all $\mathbf{S} \subseteq \mathbf{T}$, where $\mathbf{T}$ is the smaller of $\mathbf{B}(X) - \{Y\}$ and $\mathbf{B}(Y) - \{X\}$.

**[ Orient Edges ]**
    For all $X \in \mathbf{V}$ and $Y \in \mathbf{N}(X)$, orient $Y \rightarrow X$ if there exists a variable $Z \in \mathbf{N}(X) - \mathbf{N}(Y) - \{Y\}$ such that $Y$ and $Z$ are dependent given $\mathbf{S} \cup \{X\}$ for all $\mathbf{S} \subseteq \mathbf{U}$, where $\mathbf{U}$ is the smaller of $\mathbf{B}(Y) - \{Z\}$ and $\mathbf{B}(Z) - \{Y\}$.

**[ Remove Cycles ]**
    Do the following while there exist cycles in the graph:

1. Compute the set of edges $\mathbf{C} = \{X \rightarrow Y$ such that $X \rightarrow Y$ is part of a cycle$\}$.

2. Remove the edge in $\mathbf{C}$ that is part of the greatest number of cycles, and put it in $\mathbf{R}$.

**[ Reverse Edges ]**
    Insert each edge from **R** in the graph, reversed.
**[ Propagate Directions ]**
    For all $X \in \mathbf{V}$ and $Y \in \mathbf{N}(X)$ such that neither $Y \to X$ nor $X \to Y$, execute the
    following rule until it no longer applies: If there exists a directed path from $X$ to $Y$,
    orient $X \to Y$.

In the algorithm description above, step 2 determines which of the members of the blanket
of each node are actually direct neighbors (parents and children). Assuming, without loss of
generality, that $\mathbf{B}(X) - \{Y\}$ is the smaller set, if any of the tests are successful in separating
(making independent) $X$ from $Y$, the algorithm determines that there is no direct connection
between them. That would happen when the conditioning set **S** includes all parents of $X$
and no common children of $X$ and $Y$. It is interesting to note that the motivation behind
selecting the smaller set to condition on stems not only from computational efficiency
but from reliability as well: a conditioning set **S** causes the data set to be split into $2^{|\mathbf{S}|}$
partitions; smaller conditioning sets cause the data set to be split into larger partitions and
make dependence tests more reliable.

Step 3 exploits the fact that two variables that have a common descendant become dependent
when conditioning on a set that includes any such descendant. Since the direct neighbors
of $X$ and $Y$ are known from step 2, we can determine whether a direct neighbor $Y$ is a
parent of $X$ if there exists another node $Z$ (which, coincidentally, is also a parent) such
that any attempt to separate $Y$ and $Z$ by conditioning on a subset of the blanket of $Y$ that
includes $X$, fails (assuming that $\mathbf{B}(Y)$ is smaller than $\mathbf{B}(Z)$). If the directionality is indeed
$Y \to X \leftarrow Z$, there should be no such subset since, by conditioning on $X$, a permanent
dependency path between $Y$ and $Z$ is created. This would not be the case if $Y$ were a child
of $X$.

It is straightforward to show that the algorithm requires $O(n^2 + nb^2 2^b)$ conditional inde-
pendence tests, where $b = \max_X(|\mathbf{B}(X)|)$. Under the assumption that $b$ is bounded by a
constant, this algorithm is $O(n^2)$ in the number of conditional independence tests. It is
worthwhile to note that the time to compute a conditional independence test by a pass over
the data set **D** is $O(n\,|\mathbf{D}|)$ and *not* $O(2^{|\mathbf{V}|})$. An analysis and a formal proof of correctness
of the algorithm is presented in [Margaritis99].

**Discussion**

The main advantage of the algorithm comes through the use of Markov blankets to restrict
the size of the conditioning sets. The Markov blankets may be usually wrong in the side
of including too many nodes because they are represented by a disjunction of tests for
all values of the conditioning set, on the same data. This emphasizes the importance of
the "direct neighbors" step which removes nodes that were incorrectly added during the
Markov blanket computation step by admitting variables whose dependence was shown
high confidence in a large number of different tests.

It is also possible that an edge direction is wrongly determined during step 3 due to non-
representative or noisy data. This may lead to directed cycles in the resulting graph. It is
therefore necessary to remove those cycles by identifying the minimum set of edges than
need to be reversed for all cycles to disappear. This problem is closely related [Margaritis99]
to the *Minimum Feedback Arc Set* problem, which is concerned with identifying a minimum
set of edges that need to be removed from a graph that possibly contains directed cycles,
in order for all such cycles to disappear. Unfortunately, this problem is NP-complete in its
generality [Jünger85]. We introduce here a reasonable heuristic for its solution that is based
on the number of cycles that an edge that is part of a cycle is involved in.

Not all edge directions can be determined during the last two steps. For example, nodes with
a single parent or multi-parent nodes (called *colliders*) whose parents are directly connected
do not apply to step 3, and steps 4 and 5 are only concerned with already directed edges.
Step 6 attempts to ameliorate that, through orienting edges in a way that does not introduce

a cycle, if the reverse direction necessarily does. It is not obvious that, for example, if the direction $X \to Y$ produces a cycle in an otherwise acyclic graph, the opposite direction $Y \to X$ will not also. However, this is the case. For the proof of this, see [Margaritis99].

The algorithm is similar to the SGS algorithm presented in [Spirtes93], but differs in a number of ways. Its main difference lies in the use of Markov blankets to dramatically improve performance (in many cases where the bounded blanket size assumptions hold). Its structure is similar to SGS, and the stability (frequently referred to as robustness in the following discussion) arguments presented in [Spirtes93] apply. Increased reliability stems from the use of smaller conditioning sets, leading to greater number of examples per test. The PC algorithm, also in [Spirtes93], differs from the GS algorithm in that it involves linear probing for a separator set, which makes it unnecessarily inefficient.

## 4 Randomized Version of the GS Algorithm

The GS algorithm, as presented above, is appropriate for situations where the maximum Markov blanket of each of a set of variables is small. While it is reasonable to assume that in many real-life problems where high-level variables are involved this may be the case, other problems such as Bayesian image retrieval in computer vision, may employ finer representations. In these cases the variables used may depend in a direct manner on many others. For example, we may choose to use variables to characterize local texture in different parts of an image. If the resolution of the mapping from textures to variables is increasingly fine, direct dependencies among those variables may be plentiful and therefore the maximum Markov blanket size may be significant.

Another problem that has plagued independence-test based algorithms for Bayesian net structure induction in general is that their decisions are based on a single or a few tests ("hard" decisions), making them prone to errors due to noise in the data. This also applies to the the GS algorithm. It would therefore be advantageous to employ multiple tests before deciding on a direct neighbor or the direction of an edge.

The randomized version of the GS algorithm addresses these two problems. Both of them are tackled through randomized testing and Bayesian evidence accumulation. The problem of exponential running times in the maximum blanket size of steps 2 and 3 of the plain algorithm is overcome by replacing them by a series of tests, whose number may be specified by the user, with the members of the conditioning set chosen randomly from the smallest blanket of the two variables. Each such test provides evidence for or against the direct connection between the two variables, appropriately weighted by the probability that circumstances causing that event occur or not, and due to the fact that connectedness is the conjunction of more elementary events.

This version of the algorithm is not shown here in detail due to space restrictions. Its operation follows closely the one of the plain GS version. The main difference lies in the usage of Bayesian updating of the posterior probability of a direct link (or a dependence through a collider) between a pair of variables $X$ and $Y$ using conditional dependence tests that take into account independent evidence. The posterior probability $p_i$ of a link between $X$ and $Y$ after executing $i$ dependence tests $d_j, j = 1, \ldots, i$ is

$$p_i = \frac{p_{i-1} d_i}{p_{i-1} d_i + (1 - p_{i-1})(G + 1 - d_i)}$$

where $G \equiv G(X, Y) = 1 - (\frac{1}{2})^{|\mathbf{T}|}$ is a factor that takes values in the interval $[0, 1)$ and can be interpreted as the "(un)importance" of the truth of each test $d_i$, while $\mathbf{T}$ is the smaller of $\mathbf{B}(X) - \{Y\}$ and $\mathbf{B}(Y) - \{X\}$. We can use this accumulated evidence to guide our decisions to the hypothesis that we feel most confident about. Besides being able to do that in a timely manner due to the user-specified number of tests, we also note how this approach also addresses the robustness problem mentioned above through the use of multiple weighted tests, and leaving for the end the "hard" decisions that involve a threshold (*ie.* comparing the posterior probability with a threshold, which in our case is $\frac{1}{2}$).

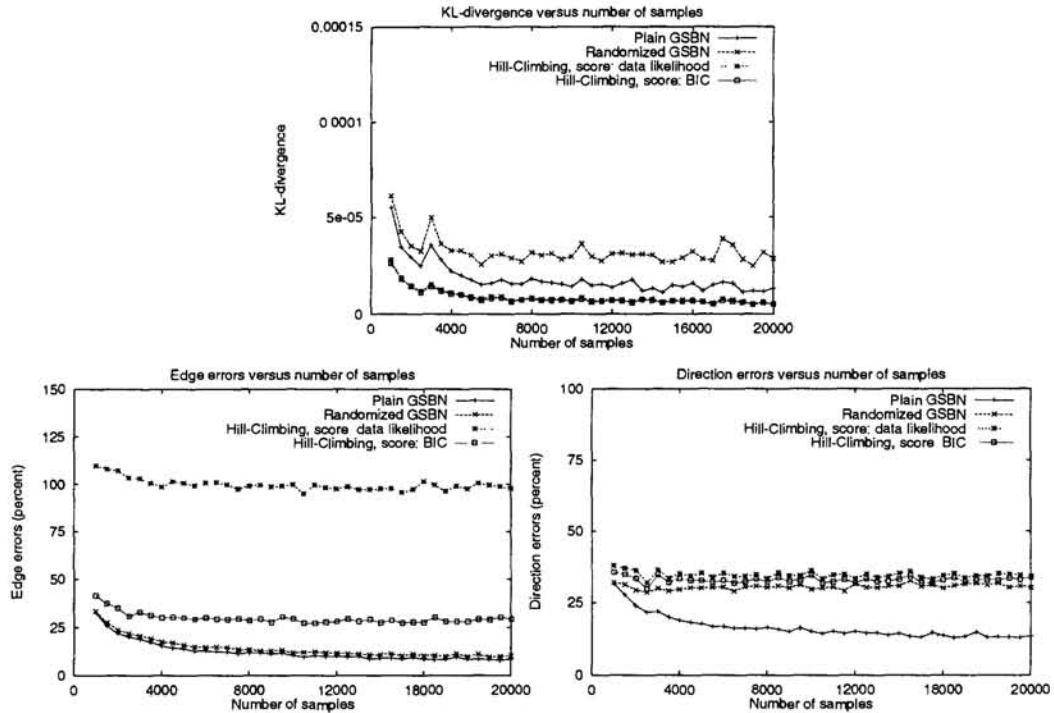

Figure 3: Results for a $5 \times 5$ rectangular net with branching factor 2 (in both directions, blanket size 8) as a function of the number of samples. On the top, KL-divergence is depicted for the plain GS, randomized GS, and hill-climbing algorithms. On the bottom, the percentage of edge and direction errors are shown. Note that certain edge error rates for the hill-climbing algorithm exceed 100%.

## 5   Results

Throughout the algorithms presented in this paper we employ standard chi-square ($\chi^2$) conditional dependence tests (as is done also in [Spirtes93]) in order to compare the histograms $\widehat{P}(X)$ and $\widehat{P}(X \mid Y)$. The $\chi^2$ test gives us a probability of the error of assuming that the two variables are dependent when in fact they are not (type II error of a dependence test), from which we can easily derive the probability that $X$ and $Y$ are dependent. There is an implicit confidence threshold $\tau$ involved in each dependence test, indicating how certain we wish to be about the correctness of the test without unduly rejecting dependent pairs, something that is always possible in reality due to the presence of noise. In all experiments we used $\tau = 0.95$, which corresponds to a 95% confidence test.

We test the effectiveness of the algorithms through the following procedure: we generate a random rectangular net of specified dimensions and up/down branching factor. A number of examples are drawn from that net using logic sampling and they are used as input to the algorithm under test. The resulting nets can be compared with the original ones along dimensions of KL-divergence and difference in edges and edge directionality. The KL-divergence was estimated using a Monte Carlo procedure. An example reconstruction was shown in the beginning of the paper, Fig. 1.

Fig. 3 shows how the KL-divergence between the original and the reconstructed net as well as edge omissions/false additions/reversals as a function of number of samples used. It demonstrates two facts. First, that typical KL-divergence for both GS and hill-climbing algorithms is low (with hill-climbing slightly lower), which shows good performance for applications where prediction is of prime concern. Second, the number of incorrect edges and the errors in the directionality of the edges present is much higher for the hill-climbing algorithm, making it unsuitable for accurate Bayesian net reconstruction.

Fig. 4 shows the effects of increasing the Markov blanket through an increasing branching factor. As expected, we see a dramatic (exponential) increase in execution time of the plain

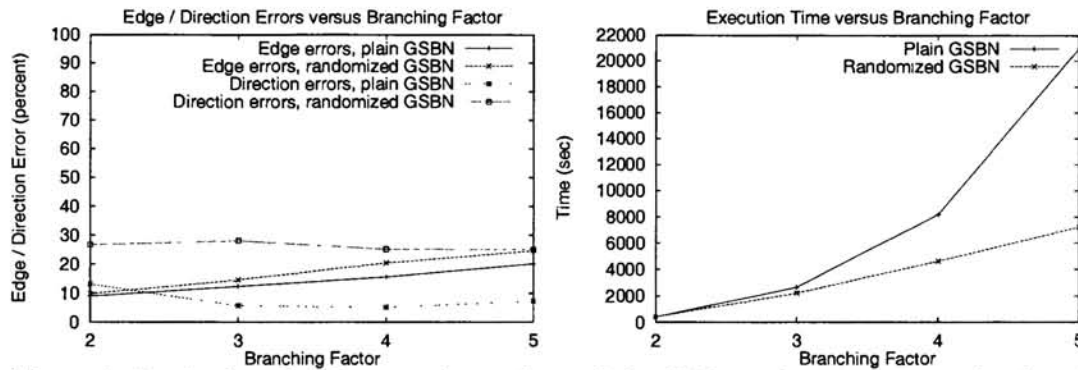

Figure 4: Results for a $5 \times 5$ rectangular net from which 10000 samples were generated and used for reconstruction, versus increasing branching factor. On the left, errors are slowly increasing as expected, but comparable for the plain and randomized versions of the GS algorithm. On the right, corresponding execution times are shown.

GS algorithm, though only a mild increase of the randomized version. The latter uses 200 (constant) conditional tests per decision, and its execution time increase can be attributed to the (quadratic) increase in the number of decisions. Note that the error percentages between the plain and the randomized version remain relatively close. The number of direction errors for the GS algorithm actually decreases due to the larger number of parents for each node (more "V" structures), which allows a greater number of opportunities to recover the directionality of an edge (using an increased number of tests).

# 6 Discussion

In this paper we presented an efficient algorithm for computing the Markov blanket of a node and then used it in the two versions of the GS algorithm (plain and randomized) by exploiting the properties of the Markov blanket to facilitate fast reconstruction of the local neighborhood around each node, under assumptions of bounded neighborhood size. We also presented a randomized variant that has the advantages of faster execution speeds and added reconstruction robustness due to multiple tests and Bayesian accumulation of evidence. Simulation results demonstrate the reconstruction accuracy advantages of the algorithms presented here over hill-climbing methods. Additional results also show that the randomized version has a dramatical execution speed benefit over the plain one in cases where the assumption of bounded neighborhood does not hold, without significantly affecting the reconstruction error rate.

# References

[Chow68]     C.K. Chow and C.N. Liu. Approximating discrete probability distributions with dependence trees. *IEEE Transactions on Information Theory*, 14, 1968.

[Herskovits90]  E.H. Herskovits and G.F. Cooper. Kutató: An entropy-driven system for construction of probabilistic expert systems from databases. UAI-90.

[Spirtes93]   P. Spirtes, C. Glymour, and R. Scheines. *Causation, Prediction, and Search*, Springer, 1993.

[Pearl88]    J. Pearl. *Probabilistic Reasoning in Intelligent Systems*, Morgan Kaufmann, 1988.

[Rebane87]    G. Rebane and J. Pearl. The recovery of causal poly-trees from statistical data. UAI-87.

[Verma90]    T.S. Verma, and J. Pearl. Equivalence and Synthesis of Causal Models. UAI-90.

[Agosta88]    J.M. Agosta. The structure of Bayes networks for visual recognition. UAI-88.

[Cheng97]    J. Cheng, D.A. Bell, W. Liu, An algorithm for Bayesian network construction from data. AI and Statistics, 1997.

[Margaritis99]  D. Margaritis, S. Thrun, Bayesian Network Induction via Local Neighborhoods. TR CMU-CS-99-134, forthcoming.

[Jünger85]    M. Jünger, *Polyhedral combinatorics and the acyclic subdigraph problem*, Heldermann, 1985.
